# Model of a Biological Neuron as a Temporal Neural Network

**Sean D. Murphy and Edward W. Kairiss**
Interdepartmental Neuroscience Program, Department of Psychology,
and The Center for Theoretical and Applied Neuroscience,
Yale University,
Box 208205, New Haven, CT 06520

## Abstract

A biological neuron can be viewed as a device that maps a multidimensional temporal event signal (dendritic postsynaptic activations) into a unidimensional temporal event signal (action potentials). We have designed a network, the Spatio-Temporal Event Mapping (STEM) architecture, which can learn to perform this mapping for arbitrary biophysical models of neurons. Such a network appropriately trained, called a STEM cell, can be used in place of a conventional compartmental model in simulations where only the transfer function is important, such as network simulations. The STEM cell offers advantages over compartmental models in terms of computational efficiency, analytical tractabililty, and as a framework for VLSI implementations of biological neurons.

## 1 INTRODUCTION

Discovery of the mechanisms by which the mammalian cerebral cortex processes and stores information is the greatest remaining challenge in the brain sciences. Numerous modeling studies have attempted to describe cortical information processing in frameworks as varied as holography, statistical physics, mass action, and nonlinear dynamics. Yet, despite these theoretical studies and extensive experimental efforts, the functional architecture of the cortex and its implementation by cortical neurons are largely a mystery.

Our view is that the most promising approach involves the study of computational models with the following key properties: (1) Networks consist of large ($>10^3$) numbers of neurons; (2) neurons are connected by modifiable synapses; and (3) the neurons themselves possess biologically-realistic dynamics.

Property (1) arises from extensive experimental observations that information processing and storage is distributed over many neurons. Cortical networks are also characterized by *sparse connectivity*: the probability that any two local cortical neurons are synaptically connected is typically less than 0.1. These and other observations suggest that key features of cortical dynamics may not be apparent unless large, sparsely-connected networks are studied.

Property (2) is suggested by the accumulated evidence that (a) memory formation is subserved by use-dependent synaptic modification, and (b) Hebbian synaptic plasticity is present in many areas of the brain thought to be important for memory. It is also well known that artificial networks composed of elements that are connected by Hebb-like synapses have powerful computational properties.

Property (3) is based on the assumption that biological neurons are computationally more complex than, for example, the processing elements that compose artificial (connectionist) neural networks. Although it has been difficult to infer the computational function of cortical neurons directly from experimental data, models of neurons that explicitly incorporate biophysical components (e.g. neuronal geometry, channel kinetics) suggest a complex, highly non-linear dynamical transfer function. Since the "testability" of a model depends on the ability to make predictions in terms of empirically measurable single-neuron firing behavior, a biologically-realistic nodal element is necessary in the network model.

Biological network models with the above properties (e.g. Wilson & Bower, 1992; Traub and Wong, 1992) have been handicapped by the computationally expensive single-neuron representation. These "compartmental" models incorporate the neuron's morphology and membrane biophysics as a large ($10^2$ -$10^4$) set of coupled, non-linear differential equations. The resulting system is often stiff and requires higher-order numerical methods and small time-steps for accurate solution. Although the result is a realistic approximation of neuronal dynamics, the computational burden precludes exhaustive study of large networks for functionality such as learning and memory.

The present study is an effort to develop a computationally efficient representation of a single neuron that does not compromise the biological dynamical behavior. We take the position that the "dynamical transfer function" of a neuron is essential to its computational abstraction, but that the underlying molecular implementation need not be explicitly represented unless it is a target of analysis. We propose that a biological neuron can be viewed as a device that performs a mapping from multidimensional spatio-temporal (synaptic) events to unidimensional temporal events (action potentials). This computational abstraction will be called a Spatio-Temporal Event Mapping (STEM) cell. We describe the architecture of the neural net that implements the neural transfer function, and the training procedure required to develop realistic dynamics. Finally, we discuss our preliminary analyses of the performance of the model when compared with the full biophysical representation.

## 2   STEM ARCHITECTURE

The architecture of the STEM cell is similar to that found in neural nets for temporal sequence processing (e.g. review by Mozer, in press). In general, these networks have 2 components: (1) a short-term memory mechanism that acts as a preprocessor for (2) a non-linear feedforward network. For example, de Vries & Principe (1992) describe the utility of the gamma net, a real-time neural net for temporal processing, in time series prediction. The preprocessor in the gamma net is the gamma memory structure, implemented as a network of adaptive dispersive elements (de Vries & Principe, 1991). The preprocessor in our model (the "tau layer", described below) is somewhat simpler, and is inspired by the temporal dynamics of membrane conductances found in biological neurons.

The STEM architecture (diagrammed in Figure 1) works by building up a vectorial representation of the state of the neuron as it continuously receives incoming synaptic activations, and then labeling that vector space as either "FIRE" or "DON'T FIRE". This is accomplished with the use of four major components: (1) TAU LAYER: a layer of nodes that continuously maps incoming synaptic activations into a finite-dimensional vector space (2) FEEDBACK TAU NODE: a node that maintains a vectorial representation of the past activity of the cell itself (3) MLP: a multilayer perceptron that functions as a non-linear spatial mapping network that performs the "FIRE" / "NO-FIRE" labeling on the tau layer output (4) OUTPUT FILTER: this adds a refractory period and threshold to the MLP output that contrains the format of the output to be discrete-time events.

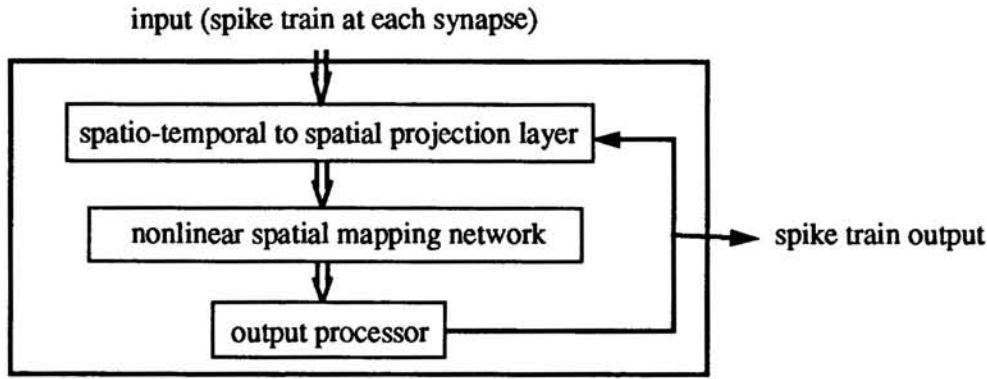

Figure 1: Information Flow in The STEM Cell

The tau layer (Fig. 2) consists of N + 1 tau nodes, where N is the number of synapses on the cell, and the extra node is used for feedback. Each tau node consists of M tau units. Each tau unit within a single tau node receives an identical input signal. Each tau unit within a tau node calculates a second-order rise-and-decay function with unique time constants. The tau units within a tau node translate arbitrary temporal events into a vector form, with each tau-unit corresponding to a different vector component. Taken as a whole, all of the tau unit outputs of the tau node layer comprise a high-dimensional vector that represents the overall state of the neuron. Functionally, the tau layer approximates a one-to-one mapping between the spatio-temporal input and the tau-unit vector space.

The output of each tau unit in the tau layer is fed into the input layer of a multilayer perceptron (MLP) which, as will be explained in the next section, has been trained to label the tau-layer vector as either FIRE or NO-FIRE. The output of the MLP is then fed into an output filter with a refractory period and threshold. The STEM architecture is illustrated in Fig. 3.

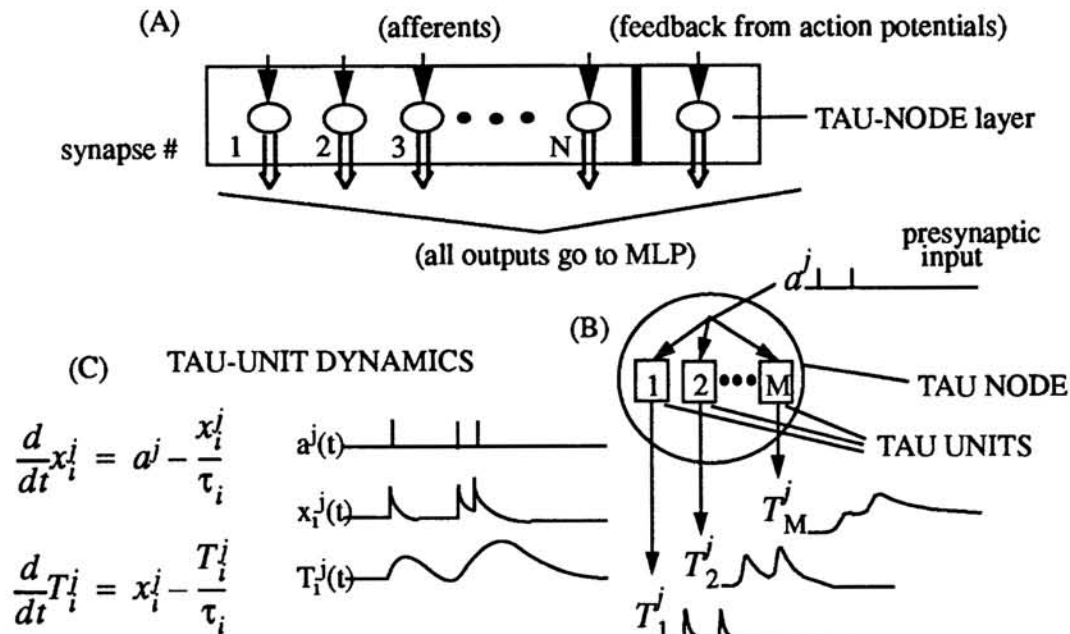

Figure 2: Tau Layer Schematic. (A) the tau layer has an afferent section, with N tau nodes, and a single-node feedback section. (B) Each tau node contains M tau units, and therefore has 1 input and M outputs (C) Each of the M tau units in a tau node has a rise and decay function with different constants. The equations are given for the ith tau unit of the jth tau node. a is input activity, x an internal state variable, and T the output.

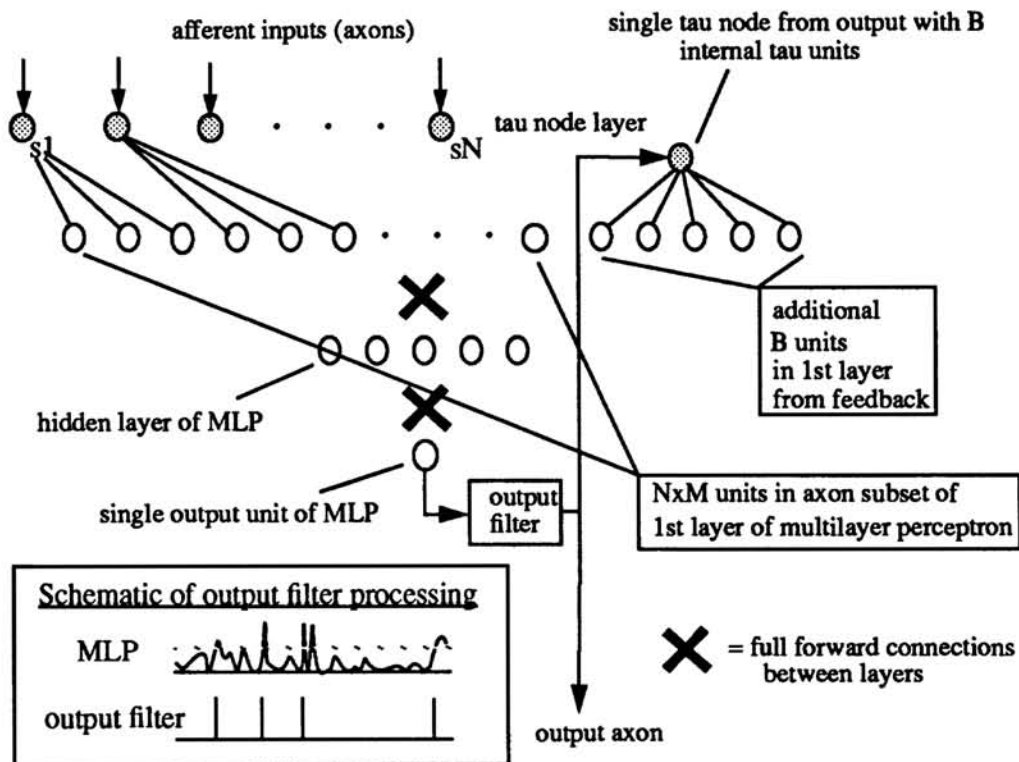

Figure 3: STEM Architecture. Afferent activity enters the tau layer, where it is converted into a vectorial representation of past spaiotemporal activity. The MLP maps this vector into a FIRE/NO-FIRE output unit, the continuous value of which is converted to a discrete event signal by the refractory period and threshold of the output filter.

## 3 STEM TRAINING

There are six phases to training the STEM cell:

(1) Biology: anatomical and physiological data are collected on the cell to be modeled.

(2) Compartmental Model: a compartmental model of the cell is designed, typically with a simulation environment such as GENESIS. As much biological detail as possible is incorporated into the model.

(3) Transfer Function Trials: many random input sequences are generated for the compartmental model. The firing response of the model is recorded for each input sequence.

(4) Sampling assignments: In the next step, sampling will need to be done on the affect of the input sequences on the STEM tay layer. The timing of the sampling is calculated by separating the response of the compartmental model on each trial into regions where no spikes occur, and regions surrounding spikes. High-rate sampling times are determined for spike regions, and lower rate times are determined for quiet regions.

(5) Tau layer trials: the identical input sequences applied to the compartmental model in step #3 are applied to an isolated tau layer of the STEM cell. The spike events from the compartmental model are used as input for the feedback node. For each input sequence, the tay layer is sampled at the times calculated in step #4, and the vector is labeled as FIRE or NO-FIRE (0 or 1).

(6) MLP training: conjugate-gradient and line-search methods are used to train the multilayer perceptron using the results of step #5 as training vectors.

Training is continued until a minimal performance level is reached, as determined by comparing the response of the STEM cell to the original compartmental model on novel input sequences.

## 4 RESULTS

The STEM cell has been initially evaluated using Roger Traub's (1991) compartmental model of a hippocampal CA1 cell, implemented in GENESIS by Dave Beeman. This is a relatively simple model structurally, with 19 compartments connected in a linear segment, with the soma in the middle. Dynamically, however, it is one of the most accurate and sophisticated models published, with on the order of 100 voltage- and Ca++ sensitive membrane channel mechanisms. 94 synapses were placed on the model. Each synapse recevied a random spike train with average frequency 10Hz during training. A diagram of the model and the locations of synaptic input is given in Fig. 4.

Inputs going to a single compartment were treated as members of a common synapse, so there were a total of 13 tau nodes, with 5 tau units per node, for a total of 65 tau units, plus 5 additional units from the feedback tau node. These fed into 70 units in the input layer of the MLP. Two STEM cells were trained, one on a passive shell of the CA1 cell, and the other with all of the membrane channels included. Both used 70 units in the hidden layer

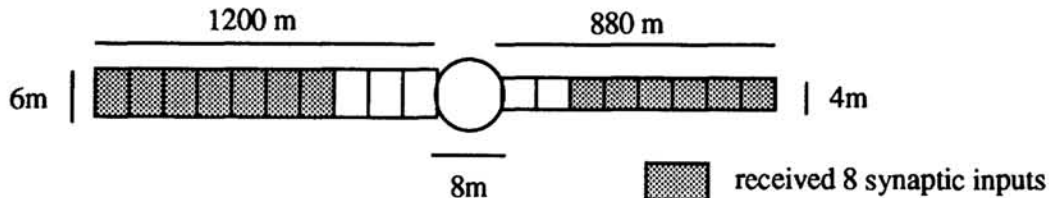

Fig. 4 Structure of Traub's CA1 cell

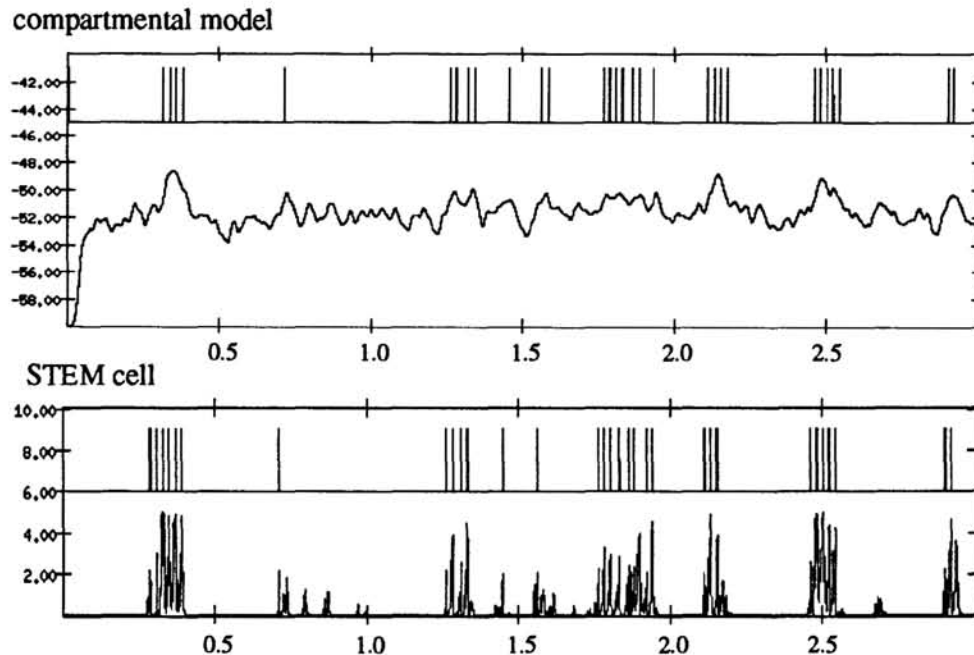

Figure 5: CA1 Passive Cell Output. The somatic voltage for the passive compartmental model and the corresponding output-filtered spike events are given in the upper graph. The lower graph shows the repsonse of the STEM cell to the same input. The upper trace of the lower graph is the output filter response, the lower trace is the raw output of the MLP. Horizontal axes in seconds. Vertical axis: top, mV, and bottom, arbitrary units.

compartmental model

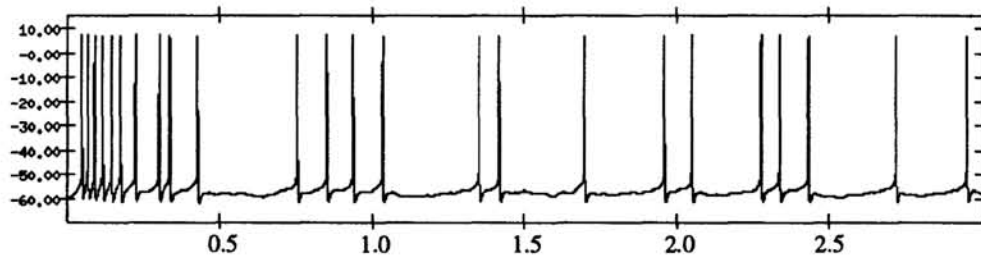

STEM cell

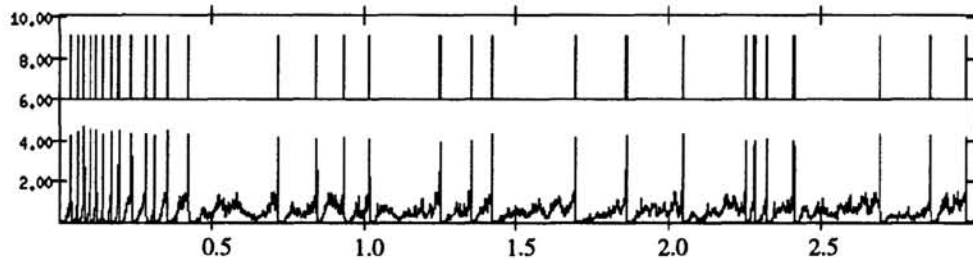

Figure 6: Response of Active CA1 Cell. The upper graph is the somatic voltage of the CA1 cell in response to a random input. The lower graph is the response of the STEM cell to the same input. The upper trace of the STEM graph is the output filter response, and the lower trace is the raw MLP output. Horizontal axes in seconds. Vertical axis: top, mV, and bottom, arbitrary units.

of the MLP. Comparisons of the compartmental model vs. STEM are shown in Fig. 5 and , for the passive and active models, respectively, Fig. 6.

For the active cell model, the STEM cell was approximately 10 times faster, and used 10% of the memory as the compartmental model.

## 5  DISCUSSION

The STEM cell is a general spatio-temporal node architecture and is similar to many context networks that have been previously developed. Its role as a single node in a large meta-network is unexplored, albeit interesting because of its capacity to mimic the transfer functions of biological neurons. Between the complexity range of connectionist networks to biological networks, there may be a multitude of useful computational schemes for solving different types of problems. The STEM cell is an attempt to efficiently capture some elements of biological neural function in a form that can be scaled along this range.

Characterization of the computational function of the neuron is a topic of considerable interest and debate. STEM cells may be useful as a rough measure of how complex the transfer function of a given biophysical model is. For example, it might be able to answer the question: Which instantiates a more sophisticated nonlinear spatio-temporal map, a single-compartment cell with complex somatic $Ca^{++}$ dynamics, or a cell with active $Na^+$ channels in a complex dendritic tree?

STEM architecture may also be interesting for theoretical and applied ANN research as a connectionist representation of a biological neuron. The expanding body of work on focused network architectures (Mozer, 1989; Stornetta, 1988) may be an avenue towards the formalization of biological neural transfer functions. Because a VLSI implementation

of a STEM cell could be reprogrammed on the fly to assume the transfer function of any pre-trained biophysically-modeled cell, a VLSI STEM network chip is a more versatile approach to artifical implementations of biological neurons than reconstructing compartmental models in VLSI.

Our future plans include using networks with delay lines and hebbian learning rules, both of which the STEM architecture is directly suited for, to investigate the capacity for STEM networks to perform real-time dynamic pattern tracking. The present implementation of the STEM cell is by no means an optimal one. We are experimenting with alternative components for the MLP, such as modular recurrent networks.

**Acknowledgements**

This work was supported by the Yale Center for Theoretical and Applied Neuroscience.

**References**

de Vries, B. and Principe, J.C. (1991) A theory for neural networks with time delays. In R.P. Lippmann, J. Moody, & D.S. Touretzky (Eds.), *Advances in Neural Information Processing Systems 3* (pp. 162-168). San Mateo, CA: Morgan Kaufmann.

de Vries, B. and Principe, J.C. (1992) The gamma model - A new neural net model for temporal processing. *Neural Networks*, 5, 565-576.

Mozer, M.C. (1989). A focused back-propagation algorithm for temporal pattern recognition, *Complex Systems*, 3, 349-381.

Mozer, M.C. (in press) Neural net architectures for temporal sequence processing. In A. Weigend & N. Gershenfeld (Eds.), *Predicting the Future and Understanding the Past*. Redwood City, CA: Addison-Wesley.

Stornetta, W.S., Hogg, T., & Huberman, B.A. (1988). A dynamical approach to temporal pattern processing. In Anderson D.Z. (Ed.), *Neural Information Processing Systems*, 750-759.

Traub, R.D. and Wong, R. K. S. (1991). *Neuronal Networks of the Hippocampus*. Cambridge: Cambridge University Press.

Wilson, M. and Bower, J.M. (1992) Cortical oscillations and temporal interactions in a computer simulation of piriform cortex. *J. Neurophysiol.* 67:981-95.

